# Estimating the Reliability of ICA Projections

**F. Meinecke[1,2], A. Ziehe[1], M. Kawanabe[1] and K.-R. Müller[1,2*]**
[1]Fraunhofer FIRST.IDA, Kekuléstr. 7, 12489 Berlin, Germany
[2]University of Potsdam, Am Neuen Palais 10, 14469 Potsdam, Germany
{meinecke,ziehe,nabe,klaus}@first.fhg.de

## Abstract

When applying unsupervised learning techniques like ICA or temporal decorrelation, a key question is whether the discovered projections are reliable. In other words: can we give *error bars* or can we assess the *quality* of our separation? We use resampling methods to tackle these questions and show experimentally that our proposed variance estimations are strongly correlated to the separation error. We demonstrate that this reliability estimation can be used to choose the appropriate ICA-model, to enhance significantly the separation performance, and, most important, to mark the components that have a actual physical meaning. Application to 49-channel-data from an magnetoencephalography (MEG) experiment underlines the usefulness of our approach.

## 1 Introduction

Blind source separation (BSS) techniques have found wide-spread use in various application domains, e.g. acoustics, telecommunication or biomedical signal processing. (see e.g. [9, 5, 6, 1, 2, 4, 14, 8]).
BSS is a statistical technique to reveal unknown source signals when only mixtures of them can be observed. In the following we will only consider linear mixtures; the goal is then to estimate those projection directions, that recover the source signals. Many different BSS algorithms have been proposed, but to our knowledge, so far, no principled attempts have been made to assess the reliability of BSS algorithms, such that error bars are given along with the resulting projection estimates. This lack of error bars or means for selecting between competing models is of course a basic dilemma for most unsupervised learning algorithms. The sources of potential unreliability of unsupervised algorithms are ubiquous, i.e. noise, non-stationarities, small sample size or inadequate modeling (e.g. sources are simply dependent instead of independent). Unsupervised projection techniques like PCA or BSS will always give an answer that is found within their model class, e.g. PCA will supply an orthogonal basis even if the correct modeling might be non-orthogonal. But how can we assess such a miss-specification or a large statistical error?
Our approach to this problem is inspired by the large body of statistics literature on

*resampling* methods (see [12] or [7] for references), where algorithms for assessing the stability of the solution have been analyzed e.g. for PCA [3].

We propose reliability estimates based on bootstrap resampling. This will enable us to *select* a good BSS *model*, in order to *improve* the separation performance and to find potentially *meaningful* projection directions. In the following we will give an algorithmic description of the resampling methods, accompanied by some theoretical remarks (section 2) and show excellent experimental results (sections 3 and 4). We conclude with a brief discussion.

## 2 Resampling Techniques for BSS

### 2.1 The ICA Model

In blind source separation we assume that at time instant $t$ each component $x_i(t)$ of the observed $n$-dimensional data vector, $\mathbf{x}(t)$ is a linear superposition of $m \leq n$ statistically independent signals:

$$x_i(t) = \sum_{j=1}^{m} A_{ij} s_j(t)$$

(e.g. [8]). The source signals $s_j(t)$ are unknown, as are the coefficients $A_{ij}$ of the mixing matrix $\mathbf{A}$. The goal is therefore to estimate both unknowns from a sample of the $\mathbf{x}(t)$, i.e. $\mathbf{y}(t) = \hat{\mathbf{s}}(t) = \mathbf{W}\mathbf{x}(t)$, where $\mathbf{W}$ is called the separating matrix.

Since both $\mathbf{A}$ and $\mathbf{s}(t)$ are unknown, it is impossible to recover the scaling or the order of the columns of the mixing matrix $\mathbf{A}$. All that one can get are the projection *directions*. The mixing/demixing process can be described as a change of coordinates. From this point of view the data vector stays the same, but is expressed in different coordinate systems (passive transformation). Let $\{\mathbf{e}_i\}$ be the canonical basis of the true sources $\mathbf{s} = \sum \mathbf{e}_i s_i$. Analogous, let $\{\mathbf{f}_j\}$ be the basis of the estimated ICA channels: $\mathbf{y} = \sum \mathbf{f}_j y_j$. Using this, we can define a component-wise separation error $E_i$ as the angle difference between the true direction of the source and the direction of the respective ICA channel:

$$E_i = \arccos\left(\frac{\mathbf{e}_i \cdot \mathbf{f}_i}{||\mathbf{e}_i|| \cdot ||\mathbf{f}_i||}\right).$$

To calculate this angle difference, remember that component-wise we have $y_j = \sum W_{jk} A_{ki} s_i$. With $\mathbf{y} = \mathbf{s}$, this leads to: $\mathbf{f}_j = \sum \mathbf{e}_i (\mathbf{WA})^{-1}_{ij}$, i.e. $\mathbf{f}_j$ is the j-th column of $(\mathbf{WA})^{-1}$.

In the following, we will illustrate our approach for two different source separation algorithms (JADE, TDSEP). JADE [4] using higher order statistics is based on the joint diagonalization of matrices obtained from 'parallel slices' of the fourth order cumulant tensor. TDSEP [14] relies on second order statistics only, enforcing temporal decorrelation between channels.

### 2.2 About Resampling

The objective of resampling techniques is to produce surrogate data sets that eventually allow to approximate the 'separation error' by a repeated estimation of the parameters of interest. The underlying mixing should of course be independent of the generation process of the surrogate data and therefore remain invariant under resampling.

The most popular resampling methods are the Jackknife and the Bootstrap (see e.g. [12, 7]) The Jackknife produces surrogate data sets by just deleting one datum each time from the original data. There are generalizations of this approach like k-fold cross-validation which delete more than one datum at a time. A more general approach is the Bootstrap. Consider a block of, say, $N$ data points. For obtaining one bootstrap sample, we draw randomly $N$ elements from the original data, i.e. some data points might occur several times, others don't occur at all in the bootstrap sample. This defines a series $\{a_t\}$ with each $a_t$ telling how often the data point $\mathbf{x}(t)$ has been drawn. Then, the separating matrix is computed on the full block and repeatedly on each of the $N$-element bootstrap samples. The variance is computed as the squared average difference between the estimate on the full block and the respective bootstrap unmixings. (These resampling methods have some desirable properties, which make them very attractive; for example, it can be shown that for iid data the bootstrap estimators of the distributions of many commonly used statistics are consistent.) It is straight forward to apply this procedure to BSS algorithms that do not use time structure; however, only a small modification is needed to take time structure into account. For example, the time lagged correlation matrices needed for TDSEP, can be obtained from $\{a_t\}$ by

$$C_{ij}(\tau) = \frac{1}{N} \sum_{t=1}^{N} a_t \cdot x_i(t) x_j(t+\tau)$$

with $\sum a_t = N$ and $a_t \in \{0, 1, 2, ...\}$.

*Other resampling methods*

Besides the Bootstrap, there are other resampling methods like the Jack-knife or cross-validation which can be understood as special cases of Bootstrap. We have tried k-fold cross-validation, which yielded very similar results to the ones reported here.

## 2.3   The Resampling Algorithm

After performing BSS, the estimated ICA-projections are used to generate surrogate data by resampling. On the whitened[1] surrogate data, the source separation algorithm is used again to estimate a rotation that separates this surrogate data. In order to compare different rotation matrices, we use the fact that the matrix representation of the rotation group $SO(N)$ can be parameterized by

$$R(\alpha) = \exp\left(\frac{1}{2} \sum_{i,j} \alpha_{ij} \mathbf{M}^{ij}\right)$$

with $(M_{ab})^{ij} = \delta_a^i \delta_b^j - \delta_a^j \delta_b^i$, where the matrices $\mathbf{M}^{ij}$ are generators of the group and the $\alpha_{ij}$ are the rotation parameters (angles) of the rotation matrix $R$. Using this parameterization we can easily compare different N-dimensional rotations by comparing the rotation parameters $\alpha_{ij}$. Since the sources are already separated, the estimated rotation matrices will be in the vicinity of the identity matrix.[2]

$\mathrm{Var}(\alpha_{ij})$ measures the instability of the separation with respect to a rotation in the $(i,j)$-plane. Since the reliability of a projection is bounded by the maximum angle variance of all rotations that affect this direction, we define the uncertainty of the $i$-th ICA-Projection as $U_i := \max_j Var(\alpha_{ij})$. Let us summarize the resampling algorithm:

1. Estimate the separating matrix $\mathbf{W}$ with some ICA algorithm. Calculate the ICA-Projections $\mathbf{y} = \mathbf{Wx}$

2. Produce $k$ surrogate data sets from $\mathbf{y}$ and whiten these data sets

3. For each surrogate data set: do BSS, producing a set of rotation matrices

4. Calculate variances of rotation parameters (angles) $\alpha_{ij}$

5. For each ICA component calculate the uncertainty $U_i = \max\limits_j Var(\alpha_{ij})$.

## 2.4 Asymptotic Considerations for Resampling

Properties of resampling methods are typically studied in the limit when the number of bootstrap samples $B \to \infty$ and the length of signal $T \to \infty$ [12]. In our case, as $B \to \infty$, the bootstrap variance estimator $U_i^*(B)$ computed from the $\alpha_{ij}^{*b}$'s converge to $U_i^*(\infty) := \max_j \mathrm{Var}_{\hat{F}}[\alpha_{ij}^*]$ where $\alpha_{ij}^*$ denotes the resampled deviation and $\hat{F}$ denotes the distribution generating it. Furthermore, if $\hat{F} \to F$, $U_i^*(\infty)$ converges to the true variance $U_i = \max_j \mathrm{Var}_F[\alpha_{ij}]$ as $T \to \infty$. This is the case, for example, if the original signal is i.i.d. in time. When the data has time structure, $\hat{F}$ does not necessarily converge to the generating distribution $F$ of the original signal anymore. Although we cannot neglect this difference completely, it is small enough to use our scheme for the purposes considered in this paper, e.g. in TDSEP, where the $\alpha_{ij}$ depend on the variation of the time-lagged covariances $C_{ij}(\tau)$ of the signals, we can show that their estimators $\hat{C}_{ij}^*(\tau)$ are unbiased:

$$E_{\hat{F}}\left[\hat{C}_{ij}^*(\tau)\right] = \hat{C}_{ij}(\tau) \overset{T \to \infty}{\longrightarrow} C_{ij}(\tau).$$

Furthermore, we can bound the difference $\Delta_{ijkl}(\tau, \nu) = \mathrm{cov}_F\left[\hat{C}_{ij}(\tau), \hat{C}_{kl}(\nu)\right] - \mathrm{cov}_{\hat{F}}\left[\hat{C}_{ij}^*(\tau), \hat{C}_{kl}^*(\nu)\right]$ between the covariance of the real matrices and their bootstrap estimators as

$$|\Delta_{ijkl}(\tau, \nu)| \leq \frac{M^2}{T} \left\{ \begin{array}{l} \left(\frac{2a^2}{1-a^2} + |\tau - \nu|\right) a^{|\tau - \nu|} \delta_{ik}\delta_{jl} \\ + \left(\frac{2a^2}{1-a^2} + (\tau + \nu)\right) a^{\tau + \nu} \delta_{jk}\delta_{il} \end{array} \right\}$$

if $\exists a < 1, M \geq 1, \quad \forall i: \quad |C_{ii}(\tau)| \leq Ma^\mu|C_{ii}(0)|$. In our experiments, however, the bias is usually found to be much smaller than this upper bound.

# 3 Experiments

## 3.1 Comparing the separation error with the uncertainty estimate

To show the practical applicability of the resampling idea to ICA, the separation error $E_i$ was compared with the uncertainty $U_i$. The separation was performed on different artificial 2D mixtures of speech and music signals and different iid data sets of the same variance. To achieve different separation qualities, white gaussian noise of different intensity has been added to the mixtures.

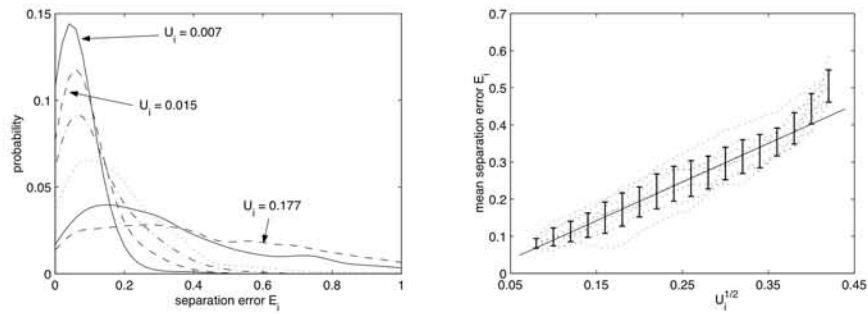

Figure 1: (a) The probability distribution for the separation error for a small uncertainty is close to zero, for higher uncertainty it spreads over a larger range. (b) The expected error increases with the uncertainty.

Figure 1 relates the uncertainty to the separation error for JADE (TDSEP results look qualitatively the same). In Fig.1 (left) we see the separation error distribution which has a strong peak for small values of our uncertainty measure, whereas for large uncertainties it tends to become flat, i.e. – as also seen from Fig.1 (right) – the uncertainty reflects very well the true separation error.

## 3.2   Selecting the appropriate BSS algorithm

As our variance estimation gives a high correlation to the (true) separation error, the next logical step is to use it as a model selection criterion for: (a) selecting some hyperparameter of the BSS algorithm, e.g. choosing the lag values for TDSEP or (b) choosing between a set of different algorithms that rely on different assumptions about the data, i.e. higher order statistics (e.g. JADE, INFOMAX, FastICA, ...) or second order statistics (e.g. TDSEP). It could, in principle, be much better to extract the first component with one and the next with another assumption/algorithm. To illustrate the usefulness of our reliability measure, we study a five-channel mixture of two channels of pure white gaussian noise, two audio signals and one channel of uniformly distributed noise. The reliability analysis for

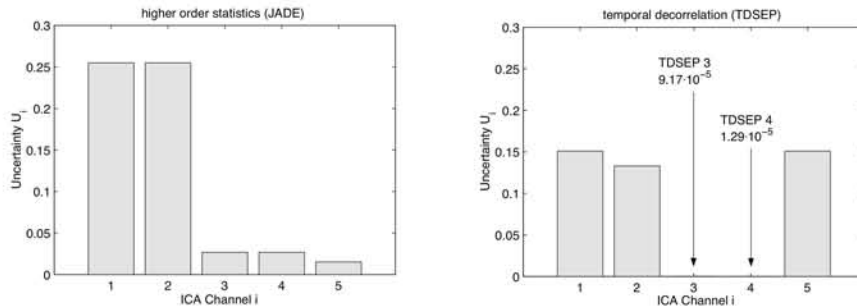

Figure 2: Uncertainty of ICA projections of an artificial mixture using JADE and TDSEP. Resampling displays the strengths and weaknesses of the different models

JADE gives the advice to rely only on channels 3,4,5 (cf. Fig.2 left). In fact, these are the channels that contain the audio signals and the uniformly distributed noise. The same analysis applied to the TDSEP-projections (time lag = 0,...,20) shows, that TDSEP can give reliable estimates only for the two audio sources (which is to be expected; cf. Fig.2 right). According to our measure, the estimation for the audio sources is more reliable in the TDSEP-case. Calculation of the separation error verifies this: TDSEP separates better by about 3 orders of magnitude (JADE:

$E_3 = 1.5 \cdot 10^{-1}, E_4 = 1.4 \cdot 10^{-1}$, TDSEP: $E_3 = 1.2 \cdot 10^{-4}, E_4 = 8.7 \cdot 10^{-5}$). Finally, in our example, estimating the audio sources with TDSEP and after this applying JADE to the orthogonal subspace, gives the optimal solution since it combines the small separation errors $E_3, E_4$ for TDSEP with the ability of JADE to separate the uniformly distributed noise.

### 3.3 Blockwise uncertainty estimates

For a longer time series it is not only important to know which ICA channels are reliable, but also to know whether different parts of a given time series are more (or less) reliable to separate than others. To demonstrate these effects, we mixed two audio sources (8kHz, 10s - 80000 data points), where the mixtures are partly corrupted by white gaussian noise. Reliability analysis is performed on windows of length 1000, shifted in steps of 250; the resulting variance estimates are smoothed. Fig.3 shows again that the uncertainty measure is nicely correlated with the true separation error, furthermore the variance goes systematically up within the noisy part but also in other parts of the time series that do not seem to match the assumptions underlying the algorithm.[3] So our reliability estimates can eventually

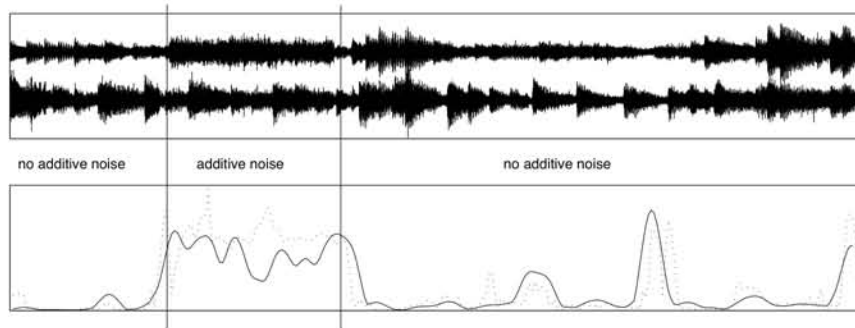

Figure 3: Upper panel: mixtures, partly corrupted by noise. Lower panel: the blockwise variance estimate (solid line) vs the true separation error on this block (dotted line).

be used to improve separation performance by removing all but the 'reliable' parts of the time series. For our example this reduces the overall separation error by 2 orders of magnitude from $2.4 \cdot 10^{-2}$ to $1.7 \cdot 10^{-4}$.

This moving-window resampling can detect instabilities of the projections in two different ways: Besides the resampling variance that can be calculated for each window, one can also calculate the change of the projection directions between two windows. The later has already been used successfully by Makeig et. al. [10].

## 4   Assigning Meaning: Application to Biomedical Data

We now apply our reliability analysis to biomedical data that has been produced by an MEG experiment with acoustic stimulation. The stimulation was achieved by presenting alternating periods of music and silence, each of 30s length, to the subjects right ear during 30 min. of total recording time (for details see [13]). The measured DC magnetic field values, sampled at a frequency of 0.4 Hz, gave a total number of 720 sample points for each of the 49 channels. While previously

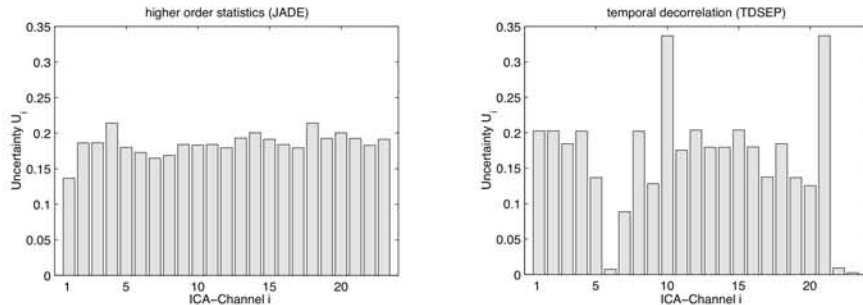

Figure 4: Resampling on the biomedical data from MEG experiment shows: (a) no JADE projection is reliable (has low uncertainty) (b) TDSEP is able to identify three sources with low uncertainty.

the JADE-projections (left) have small variance whereas TDSEP (right) identifies three sources with a good reliability. In fact, these three components have physical meaning: while component 23 is an internal very low frequency signal (drift) that is always present in DC-measurements, component 22 turns out to be an artifact of the measurement; interestingly component 6 shows a (noisy) rectangular waveform that clearly displays the $1/30s$ on/off characteristics of the stimulus (correlation to stimulus 0.7; see Fig.5). The clear dipole-structure of the spatial field pattern in

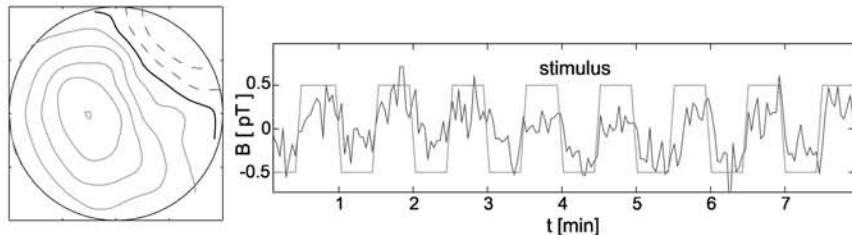

Figure 5: Spatial field pattern, frequency content and time course of TDSEP channel 6.

Fig.5 underlines the relevance of this projection. The components found by JADE do not show such a clear structure and the strongest correlation of any component to the stimulus is about 0.3, which is of the same order of magnitude as the strongest correlated PCA-component before applying JADE.

## 5 Discussion

We proposed a simple method to estimate the reliability of ICA projections based on resampling techniques. After showing that our technique approximates the separation error, several directions are open(ed) for applications. First, we may like to use it for model selection purposes to distinguish between algorithms or to chose appropriate hyperparameter values (possibly even component-wise). Second, variances

can be estimated on blocks of data and separation performance can be enhanced by using only low variance blocks where the model matches the data nicely. Finally reliability estimates can be used to find meaningful components. Here our assumption is that the more meaningful a component is, the more stably we should be able to estimate it. In this sense artifacts appear of course also as meaningful, whereas noisy directions are discarded easily, due to their high uncertainty.

Future research will focus on applying resampling techniques to other unsupervised learning scenarios. We will also consider Bayesian modelings where often a variance estimate comes for free, along with the trained model.

**Acknowledgments** K.-R.M thanks Guido Nolte and the members of the Oberwolfach Seminar September 2000 in particular Lutz Dümbgen and Enno Mammen for helpful discussions and suggestions. K. -R. M and A. Z. acknowledge partial funding by the EU project (IST-1999-14190 – BLISS). We thank the Biomagnetism Group of the Physikalisch-Technische Bundesanstalt (PTB) for providing the MEG-DC data.

## Footnotes

*To whom correspondence should be addressed.

[1]The whitening transformation is defined as $\mathbf{x}' = \mathbf{V}\mathbf{x}$ with $\mathbf{V} = E[\mathbf{x}\mathbf{x}^T]^{-1/2}$.

[2]It is important to perform the resampling when the sources are already separated, so that the $\alpha_{ij}$ are distributed around zero, because $SO(N)$ is a non-Abelian group; that means that in general $R(\alpha)R(\beta) \neq R(\beta)R(\alpha)$.

[3]For example, the peak in the last third of the time series can be traced back to the fact that the original time series are correlated in this region.

[13] analysing the data, we found that many of the ICA components are seemingly meaningless and it took some medical knowledge to find potential meaningful projections for a later close inspection. However, our reliability assessment can also be seen as indication for meaningful projections, i.e. *meaningful* components should have *low* variance. In the experiment, BSS was performed on the 23 most powerful principal components using (a) higher order statistics (JADE) and (b) temporal decorrelation (TDSEP, time lag 0..50). The results in Fig.4 show that none of

## References

[1] S. Amari, A. Cichocki, and H. H. Yang. A new learning algorithm for blind signal separation. In D.S. Touretzky, M.C. Mozer, and M.E. Hasselmo, editors, *Advances in Neural Information Processing Systems (NIPS 95)*, volume 8, pages 882–893. The MIT Press, 1996.

[2] A. J. Bell and T. J. Sejnowski. An information maximisation approach to blind separation and blind deconvolution. *Neural Computation*, 7:1129–1159, 1995.

[3] R. Beran and M.S. Srivastava. Bootstrap tests and confidence regions for functions of a covariance matrix. *Annals of Statistics*, 13:95–115, 1985.

[4] J.-F. Cardoso and A. Souloumiac. Blind beamforming for non Gaussian signals. *IEEE Proceedings-F*, 14O(6):362–370, December 1994.

[5] P. Comon. Independent component analysis, a new concept? *Signal Processing*, 36(3):287–314, 1994.

[6] G. Deco and D. Obradovic. *An information-theoretic approach to neural computing.* Springer, New York, 1996.

[7] B. Efron and R.J. Tibshirani. *An Introduction to the Bootstrap.* Chapman & Hall, first edition, 1993.

[8] A. Hyvärinen, J. Karhunen, and E. Oja. *Independent Component Analysis.* Wiley, 2001.

[9] Ch. Jutten and J. Herault. Blind separation of sources, part I: An adaptive algorithm based on neuromimetic architecture. *Signal Processing*, 24:1–10, 1991.

[10] S. Makeig, S. Enghoff, T.-P. Jung, and T. Sejnowski. Moving-window ICA decomposition of EEG data reveals event-related changes in oscillatory brain activity. In *Proc. 2nd Int. Workshop on Independent Component Analysis and Blind Source Separation (ICA'2000)*, pages 627–632, Helsinki, Finland, 2000.

[11] F. Meinecke, A. Ziehe, M. Kawanabe, and K.-R. Müller. Assessing reliability of ica projections - a resampling approach. In *ICA '01.* T.-W. Lee, Ed., 2001.

[12] J. Shao and D. Tu. *The Jackknife and Bootstrap.* Springer, New York, 1995.

[13] G. Wübbeler, A. Ziehe, B.-M. Mackert, K.-R. Müller, L. Trahms, and G. Curio. Independent component analysis of non-invasively recorded cortical magnetic dc-fields in humans. *IEEE Transactions on Biomedical Engineering*, 47(5):594–599, 2000.

[14] A. Ziehe and K.-R. Müller. TDSEP – an efficient algorithm for blind separation using time structure. In L. Niklasson, M. Bodén, and T. Ziemke, editors, *Proc. Int. Conf. on Artificial Neural Networks (ICANN'98)*, pages 675 – 680, Skövde, Sweden, 1998. Springer Verlag.